# Performance analysis for $L_2$ kernel classification

**JooSeuk Kim**
Department of EECS
University of Michigan
Ann Arbor, MI, USA
stannum@umich.edu

**Clayton D. Scott**[*]
Department of EECS
University of Michigan
Ann Arbor, MI, USA
clayscot@umich.edu

## Abstract

We provide statistical performance guarantees for a recently introduced kernel classifier that optimizes the $L_2$ or integrated squared error (ISE) of a difference of densities. The classifier is similar to a support vector machine (SVM) in that it is the solution of a quadratic program and yields a sparse classifier. Unlike SVMs, however, the $L_2$ kernel classifier does not involve a regularization parameter. We prove a distribution free concentration inequality for a cross-validation based estimate of the ISE, and apply this result to deduce an oracle inequality and consistency of the classifier on the sense of both ISE and probability of error. Our results also specialize to give performance guarantees for an existing method of $L_2$ kernel density estimation.

## 1 Introduction

In the binary classification problem we are given realizations $(\mathbf{x}_1, y_1), \ldots, (\mathbf{x}_n, y_n)$ of a jointly distributed pair $(\mathbf{X}, Y)$, where $\mathbf{X} \in \mathbb{R}^d$ is a pattern and $Y \in \{-1, +1\}$ is a class label. The goal of classification is to build a classifier, i.e., a function taking $\mathbf{X}$ as input and outputting a label, such that some measure of performance is optimized. Kernel classifiers [1] are an important family of classifiers that have drawn much recent attention for their ability to represent nonlinear decision boundaries and to scale well with increasing dimension $d$. A kernel classifier (without offset) has the form

$$g(\mathbf{x}) = \text{sign} \left\{ \sum_{i=1}^{n} \alpha_i y_i k(\mathbf{x}, \mathbf{x}_i) \right\},$$

where $\alpha_i$ are parameters and $k$ is a kernel function. For example, support vector machines (SVMs) without offset have this form [2], as does the standard kernel density estimate (KDE) plug-in rule.

Recently Kim and Scott [3] introduced an $L_2$ or integrated squared error (ISE) criterion to design the coefficients $\alpha_i$ of a kernel classifier with Gaussian kernel. Their $L_2$ classifier performs comparably to existing kernel methods while possesing a number of desirable properties. Like the SVM, $L_2$ kernel classifiers are the solutions of convex quadratic programs that can be solved efficiently using standard decomposition algorithms. In addition, the classifiers are sparse, meaning most of the coefficients $\alpha_i = 0$, which has advantages for representation and evaluation efficiency. Unlike the SVM, however, there are no free parameters to be set by the user except the kernel bandwidth parameter.

In this paper we develop statistical performance guarantees for the $L_2$ kernel classifier introduced in [3]. The linchpin of our analysis is a new concentration inequality bounding the deviation of a cross-validation based ISE estimate from the true ISE. This bound is then applied to prove an oracle inequality and consistency in both ISE and probability of error. In addition, as a special case of

---

[*]Both authors supported in part by NSF Grant CCF-0830490

our analysis, we are able to deduce performance guarantees for the method of $L_2$ kernel density estimation described in [4, 5].

The ISE criterion has a long history in the literature on bandwidth selection for kernel density estimation [6] and more recently in parametric estimation [7]. The use of ISE for optimizing the weights of a KDE via quadratic programming was first described in [4] and later rediscovered in [5]. In [8], an $\ell_1$ penalized ISE criterion was used to aggregate a finite number of pre-determined densities. Linear and convex aggregation of densities, based on an $L_2$ criterion, are studied in [9], where the densities are based on a finite dictionary or an independent sample. In contrast, our proposed method allows data-adaptive kernels, and does not require and independent (holdout) sample.

In classification, some connections relating SVMs and ISE are made in [10], although no new algorithms are proposed. Finally, the "difference of densities" perspective has been applied to classification in other settings by [11], [12], and [13]. In [11] and [13], a difference of densities are used to find smoothing parameters or kernel bandwidths. In [12], conditional densities are chosen among a parameterized set of densities to maximize the average (bounded) density differences.

Section 2 reviews the $L_2$ kernel classifier, and presents a slight modification needed for our analysis. Our results are presented in Section 3. Conclusions are offered in the final section, and proofs are gathered in an appendix.

## 2   $L_2$ **Kernel Classification**

We review the previous work of Kim & Scott [3] and introduce an important modification. For convenience, we relabel $Y$ so that it belongs to $\{1, -\gamma\}$ and denote $I_+ = \{i \mid Y_i = +1\}$ and $I_- = \{i \mid Y_i = -\gamma\}$. Let $f_-(\mathbf{x})$ and $f_+(\mathbf{x})$ denote the class-conditional densities of the pattern given the label. From decision theory, the optimal classifier has the form

$$g^*(x) = \text{sign}\left\{f_+(\mathbf{x}) - \gamma f_-(\mathbf{x})\right\}, \tag{1}$$

where $\gamma$ incorporates prior class probabilities and class-conditional error costs (in the Bayesian setting) or a desired tradeoff between false positives and false negatives [14]. Denote the "difference of densities" $d_\gamma(\mathbf{x}) := f_+(\mathbf{x}) - \gamma f_-(\mathbf{x})$.

The class-conditional densities are modelled using the Gaussian kernel as

$$\widehat{f}_+(\mathbf{x}; \boldsymbol{\alpha}) = \sum_{i \in I_+} \alpha_i k_\sigma(\mathbf{x}, \mathbf{X}_i), \quad \widehat{f}_-(\mathbf{x}; \boldsymbol{\alpha}) = \sum_{i \in I_-} \alpha_i k_\sigma(\mathbf{x}, \mathbf{X}_i)$$

with constraints $\boldsymbol{\alpha} = (\alpha_1, \ldots, \alpha_n) \in A$ where

$$A = \left\{\boldsymbol{\alpha} \mid \sum_{i \in I_+} \alpha_i = \sum_{i \in I_-} \alpha_i = 1, \quad \alpha_i \geq 0 \quad \forall i\right\}.$$

The Gaussian kernel is defined as

$$k_\sigma(\mathbf{x}, \mathbf{X}_i) = \left(2\pi\sigma^2\right)^{-d/2} \exp\left\{-\frac{\|\mathbf{x} - \mathbf{X}_i\|^2}{2\sigma^2}\right\}.$$

The ISE associated with $\boldsymbol{\alpha}$ is

$$ISE(\boldsymbol{\alpha}) = \|\widehat{d}_\gamma(\mathbf{x}; \boldsymbol{\alpha}) - d_\gamma(\mathbf{x})\|^2_{L_2} = \int \left(\widehat{d}_\gamma(\mathbf{x}; \boldsymbol{\alpha}) - d_\gamma(\mathbf{x})\right)^2 d\mathbf{x}$$

$$= \int \widehat{d}_\gamma^2(\mathbf{x}; \boldsymbol{\alpha}) d\mathbf{x} - 2\int \widehat{d}_\gamma(\mathbf{x}; \boldsymbol{\alpha}) d_\gamma(\mathbf{x}) d\mathbf{x} + \int d_\gamma^2(\mathbf{x}) d\mathbf{x}.$$

Since we do not know the true $d_\gamma(\mathbf{x})$, we need to estimate the second term in the above equation

$$H(\boldsymbol{\alpha}) \triangleq \int \widehat{d}_\gamma(\mathbf{x}; \boldsymbol{\alpha}) d_\gamma(\mathbf{x}) d\mathbf{x} \tag{2}$$

by $H_n(\boldsymbol{\alpha})$ which will be explained in detail in Section 2.1. Then, the empirical ISE is

$$\widehat{ISE}(\boldsymbol{\alpha}) = \int \widehat{d}_\gamma^2(\mathbf{x}; \boldsymbol{\alpha}) d\mathbf{x} - 2H_n(\boldsymbol{\alpha}) + \int d_\gamma^2(\mathbf{x}) d\mathbf{x}. \tag{3}$$

Now, $\widehat{\alpha}$ is defined as

$$\widehat{\alpha} = \arg\min_{\alpha \in A} \widehat{ISE}(\alpha) \tag{4}$$

and the final classifier will be

$$g(\mathbf{x}) = \begin{cases} +1, & \widehat{d}_\gamma(\mathbf{x}; \widehat{\alpha}) \geq 0 \\ -\gamma, & \widehat{d}_\gamma(\mathbf{x}; \widehat{\alpha}) < 0. \end{cases}$$

## 2.1 Estimation of $H(\alpha)$

In this section, we propose a method of estimating $H(\alpha)$ in (2). The basic idea is to view $H(\alpha)$ as an expectation and estimate it using a sample average. In [3], the resubstitution estimator for $H(\alpha)$ was used. However, since this estimator is biased, we use a leave-one-out cross-validation (LOOCV) estimator, which is unbiased and facilitates our theoretical analysis. Note that the difference of densities can be expressed as

$$\widehat{d}_\gamma(\mathbf{x}; \alpha) = \widehat{f}_+(\mathbf{x}) - \gamma\widehat{f}_-(\mathbf{x}) = \sum_{i=1}^n \alpha_i Y_i k_\sigma(\mathbf{x}, \mathbf{X}_i).$$

Then,

$$\begin{aligned} H(\alpha) &= \int \widehat{d}_\gamma(\mathbf{x}; \alpha) \, d_\gamma(\mathbf{x}) \, d\mathbf{x} = \int \widehat{d}_\gamma(\mathbf{x}; \alpha) f_+(\mathbf{x}) \, d\mathbf{x} - \gamma \int \widehat{d}_\gamma(\mathbf{x}; \alpha) f_-(\mathbf{x}) \, d\mathbf{x} \\ &= \int \sum_{i=1}^n \alpha_i Y_i k_\sigma(\mathbf{x}, \mathbf{X}_i) f_+(\mathbf{x}) \, d\mathbf{x} - \gamma \int \sum_{i=1}^n \alpha_i Y_i k_\sigma(\mathbf{x}, \mathbf{X}_i) f_-(\mathbf{x}) \, d\mathbf{x} \\ &= \sum_{i=1}^n \alpha_i Y_i h(\mathbf{X}_i) \end{aligned}$$

where

$$h(\mathbf{X}_i) \triangleq \int k_\sigma(\mathbf{x}, \mathbf{X}_i) f_+(\mathbf{x}) \, d\mathbf{x} - \gamma \int k_\sigma(\mathbf{x}, \mathbf{X}_i) f_-(\mathbf{x}) \, d\mathbf{x}. \tag{5}$$

We estimate each $h(\mathbf{X}_i)$ in (5) for $i = 1, \ldots, n$ using leave-one-out cross-validation

$$\widehat{h}_i \triangleq \begin{cases} \dfrac{1}{N_+ - 1} \sum_{j \in I_+, j \neq i} k_\sigma(\mathbf{X}_j, \mathbf{X}_i) - \dfrac{\gamma}{N_-} \sum_{j \in I_-} k_\sigma(\mathbf{X}_j, \mathbf{X}_i), & i \in I_+ \\ \dfrac{1}{N_+} \sum_{j \in I_+} k_\sigma(\mathbf{X}_j, \mathbf{X}_i) - \dfrac{\gamma}{N_- - 1} \sum_{j \in I_-, j \neq i} k_\sigma(\mathbf{X}_j, \mathbf{X}_i), & i \in I_- \end{cases}$$

where $N_+ = |I_+|$, $N_- = |I_-|$. Then, the estimate of $H(\alpha)$ is $H_n(\alpha) = \sum_{i=1}^n \alpha_i Y_i \widehat{h}_i$.

## 2.2 Optimization

The optimization problem (4) can be formulated as a quadratic program. The first term in (3) is

$$\begin{aligned} \int \widehat{d}_\gamma^2(\mathbf{x}; \alpha) \, d\mathbf{x} &= \int \left( \sum_{i=1}^n \alpha_i Y_i k_\sigma(\mathbf{x}, \mathbf{X}_i) \right)^2 d\mathbf{x} \\ &= \sum_{i=1}^n \sum_{j=1}^n \alpha_i \alpha_j Y_i Y_j \int k_\sigma(\mathbf{x}, \mathbf{X}_i) k_\sigma(\mathbf{x}, \mathbf{X}_j) \, d\mathbf{x} = \sum_{i=1}^n \sum_{j=1}^n \alpha_i \alpha_j Y_i Y_j k_{\sqrt{2}\sigma}(\mathbf{X}_i, \mathbf{X}_j) \end{aligned}$$

by the convolution theorem for Gaussian kernels [15]. As we have seen in Section 2.1, the second term $H_n(\alpha)$ in (3) is linear in $\alpha$ and can be expressed as $\sum_{i=1}^n \alpha_i c_i$ where $c_i = Y_i \widehat{h}_i$. Finally, since the third term does not depend on $\alpha$, the optimization problem (4) becomes the following quadratic program (QP)

$$\widehat{\alpha} = \arg\min_{\alpha \in A} \quad \frac{1}{2} \sum_{i=1}^n \sum_{j=1}^n \alpha_i \alpha_j Y_i Y_j k_{\sqrt{2}\sigma}(\mathbf{X}_i, \mathbf{X}_j) - \sum_{i=1}^n c_i \alpha_i. \tag{6}$$

The QP (6) is similar to the dual QP of the 2-norm SVM with hinge loss [2] and can be solved by a variant of the Sequential Minimal Optimization (SMO) algorithm [3].

# 3 Statistical performance analysis

In this section, we give theoretical performance analysis on our proposed method. We assume that $\{\mathbf{X}_i\}_{i \in I_+}$ and $\{\mathbf{X}_i\}_{i \in I_-}$ are i.i.d samples from $f_+ (\mathbf{x})$ and $f_- (\mathbf{x})$, respectively, and treat $N_+$ and $N_-$ as deterministic variables $n_+$ and $n_-$ such that $n_+ \to \infty$ and $n_- \to \infty$ as $n \to \infty$.

## 3.1 Concentration inequality for $H_n (\boldsymbol{\alpha})$

**Lemma 1.** *Conditioned on* $\mathbf{X}_i$, $\widehat{h}_i$ *is an unbiased estimator of* $h (\mathbf{X}_i)$, *i.e,*

$$\mathbf{E}\left[\widehat{h}_i \mid \mathbf{X}_i\right] = h (\mathbf{X}_i).$$

*Furthermore, for any* $\epsilon > 0$,

$$\mathbf{P}\left\{\sup_{\boldsymbol{\alpha} \in A} |H_n (\boldsymbol{\alpha}) - H (\boldsymbol{\alpha})| > \epsilon\right\} \le 2n \left(e^{-c(n_+ - 1)\epsilon^2} + e^{-c(n_- - 1)\epsilon^2}\right)$$

*where* $c = 2 \left(\sqrt{2\pi}\sigma\right)^{2d} / (1 + \gamma)^4$.

Lemma 1 implies that $H_n (\boldsymbol{\alpha}) \to H (\boldsymbol{\alpha})$ almost surely for all $\boldsymbol{\alpha} \in A$ simultaneously, provided that $\sigma$, $n_+$, and $n_-$ evolve as functions of $n$ such that $n_+ \sigma^{2d} / \ln n \to \infty$ and $n_- \sigma^{2d} / \ln n \to \infty$.

## 3.2 Oracle Inequality

Next, we establish on oracle inequality, which relates the performance of our estimator to that of the best possible kernel classifier.

**Theorem 1.** *Let* $\epsilon > 0$ *and set* $\delta = \delta (\epsilon) = 2n \left(e^{-c(n_+ - 1)\epsilon^2} + e^{-c(n_- - 1)\epsilon^2}\right)$ *where* $c = 2 \left(\sqrt{2\pi}\sigma\right)^{2d} / (1 + \gamma)^4$. *Then, with probability at least* $1 - \delta$

$$ISE (\widehat{\boldsymbol{\alpha}}) \le \inf_{\boldsymbol{\alpha} \in A} ISE (\boldsymbol{\alpha}) + 4\epsilon.$$

*Proof.* From Lemma 1, with probability at least $1 - \delta$

$$\left|ISE (\boldsymbol{\alpha}) - \widehat{ISE} (\boldsymbol{\alpha})\right| \le 2\epsilon, \quad \forall \boldsymbol{\alpha} \in A$$

by using the fact $ISE (\boldsymbol{\alpha}) - \widehat{ISE} (\boldsymbol{\alpha}) = 2 (H_n (\boldsymbol{\alpha}) - H (\boldsymbol{\alpha}))$. Then, with probability at least $1 - \delta$, for all $\boldsymbol{\alpha} \in A$, we have

$$ISE (\widehat{\boldsymbol{\alpha}}) \le \widehat{ISE} (\widehat{\boldsymbol{\alpha}}) + 2\epsilon \le \widehat{ISE} (\boldsymbol{\alpha}) + 2\epsilon \le ISE (\boldsymbol{\alpha}) + 4\epsilon$$

where the second inequality holds from the definition of $\widehat{\boldsymbol{\alpha}}$. This proves the theorem. □

## 3.3 ISE consistency

Next, we have a theorem stating that $ISE (\widehat{\boldsymbol{\alpha}})$ converges to zero in probability.

**Theorem 2.** *Suppose that for* $f = f_+$ *and* $f_-$, *the Hessian* $\mathcal{H}_f (\mathbf{x})$ *exists and each entry of* $\mathcal{H}_f (\mathbf{x})$ *is piecewise continuous and square integrable. If* $\sigma$, $n_+$, *and* $n_-$ *evolve as functions of* $n$ *such that* $\sigma \to 0$, $n_+ \sigma^{2d} / \ln n \to \infty$, *and* $n_- \sigma^{2d} / \ln n \to \infty$, *then* $ISE (\widehat{\boldsymbol{\alpha}}) \to 0$ *in probability as* $n \to \infty$

This result intuitively follows from the oracle inequality since the standard Parzen window density estimate is consistent and uniform weights are among the simplex $A$. The rigorous proof is omitted due to space limitations.

### 3.4 Bayes Error Consistency

In classification, we are ultimately interested in minimizing the probability of error. Let us now assume $\{\mathbf{X}_i\}_{i=1}^n$ is an i.i.d sample from $f(\mathbf{x}) = pf_+(\mathbf{x}) + (1-p)f_-(\mathbf{x})$, where $0 < p < 1$ is the prior probability of the positive class. The consistency with respect to the probability of error could be easily shown if we set $\gamma$ to $\gamma^* = \frac{1-p}{p}$ and apply Theorem 3 in [17]. However, since $p$ is unknown, we must estimate $\gamma^*$. Note that $N_+$ and $N_-$ are binomial random variables, and we may estimate $\gamma^*$ as $\gamma = \frac{N_-}{N_+}$. The next theorem says the $L_2$ kernel classifier is consistent with respect to the probability of error.

**Theorem 3.** *Suppose that the assumptions in Theorem 2 are satisfied. In addition, suppose that $f_- \in L_2(\mathbb{R})$, i.e. $\|f_-\|_{L_2} < \infty$. Let $\gamma = N_-/N_+$ be an estimate of $\gamma^* = \frac{1-p}{p}$. If $\sigma$ evolves as a function of $n$ such that $\sigma \to 0$ and $n\sigma^{2d}/\ln n \to \infty$ as $n \to \infty$, then the $L_2$ kernel classifier is consistent. In other words, given training data $\mathbf{D}_n = ((\mathbf{X}_1, Y_1), \dots, (\mathbf{X}_n, Y_n))$, the classification error*

$$L_n = \mathbf{P}\left\{ sgn\left\{ \widehat{d}_\gamma(\mathbf{X}; \widehat{\boldsymbol{\alpha}}) \right\} \neq Y \mid \mathbf{D}_n \right\}$$

*converges to the Bayes error $L^*$ in probability as $n \to \infty$.*

The proof is given in Appendix A.2.

### 3.5 Application to density estimation

By setting $\gamma = 0$, our goal becomes estimating $f_+$ and we recover the $L_2$ kernel density estimate of [4, 5] using leave-one-out cross-validation. Given an i.i.d sample $\mathbf{X}_1, \dots, \mathbf{X}_n$ from $f(\mathbf{x})$, the $L_2$ kernel density estimate of $f(\mathbf{x})$ is defined as

$$\widehat{f}(\mathbf{x}; \widehat{\boldsymbol{\alpha}}) = \sum_{i=1}^n \widehat{\alpha}_i k_\sigma(\mathbf{x}, \mathbf{X}_i)$$

with $\widehat{\alpha}_i$'s optimized such that

$$\widehat{\boldsymbol{\alpha}} = \underset{\substack{\sum \alpha_i = 1 \\ \alpha_i \geq 0}}{\arg\min} \frac{1}{2} \sum_{i=1}^n \sum_{j=1}^n \alpha_i \alpha_j k_{\sqrt{2}\sigma}(\mathbf{X}_i, \mathbf{X}_j) - \sum_{i=1}^n \alpha_i \left( \frac{1}{n-1} \sum_{j \neq i} k_\sigma(\mathbf{X}_i, \mathbf{X}_j) \right).$$

Our concentration inequality, oracle inequality, and $L_2$ consistency result immediately extend to provide the same performance guarantees for this method. For example, we state the following corollary.

**Corollary 1.** *Suppose that the Hessian $\mathcal{H}_f(\mathbf{x})$ of a density function $f(\mathbf{x})$ exists and each entry of $\mathcal{H}_f(\mathbf{x})$ is piecewise continuous and square integrable. If $\sigma \to 0$ and $n\sigma^{2d}/\ln n \to \infty$ as $n \to \infty$, then*

$$\int \left( \widehat{f}(\mathbf{x}; \widehat{\boldsymbol{\alpha}}) - f(\mathbf{x}) \right)^2 d\mathbf{x} \to 0$$

*in probability.*

## 4 Conclusion

Through the development of a novel concentration inequality, we have established statistical performance guarantees on a recently introduced $L_2$ kernel classifier. We view the relatively clean analysis of this classifier as an attractive feature relative to other kernel methods. In future work, we hope to invoke the full power of the oracle inequality to obtain adaptive rates of convergence, and consistency for $\sigma$ not necessarily tending to zero.

# A  Appendix

## A.1  Proof of Lemma 1

Note that for any given $i$, $(k_\sigma(\mathbf{X}_j, \mathbf{X}_i))_{j \neq i}$ are independent and bounded by $M = 1/(\sqrt{2\pi}\sigma)^d$. For random vectors $\mathbf{Z} \sim f_+(\mathbf{x})$ and $\mathbf{W} \sim f_-(\mathbf{x})$, $h(\mathbf{X}_i)$ in (5) can be expressed as

$$h(\mathbf{X}_i) = \mathbf{E}[k_\sigma(\mathbf{Z}, \mathbf{X}_i) \mid \mathbf{X}_i] - \gamma \mathbf{E}[k_\sigma(\mathbf{W}, \mathbf{X}_i) \mid \mathbf{X}_i].$$

Since $\mathbf{X}_i \sim f_+(\mathbf{x})$ for $i \in I_+$ and $\mathbf{X}_i \sim f_-(\mathbf{x})$ for $i \in I_-$, it can be easily shown that

$$\mathbf{E}\left[\widehat{h}_i \mid \mathbf{X}_i\right] = h(\mathbf{X}_i).$$

For $i \in I_+$,

$$
\begin{aligned}
\mathbf{P}&\left\{\left|\widehat{h}_i - h(\mathbf{X}_i)\right| > \epsilon \,\middle|\, \mathbf{X}_i = \mathbf{x}\right\} \\
&\leq \mathbf{P}\left\{\left|\frac{1}{n_+ - 1}\sum_{j \in I_+, j \neq i} k_\sigma(\mathbf{X}_j, \mathbf{X}_i) - \mathbf{E}[k_\sigma(\mathbf{Z}, \mathbf{X}_i) \mid \mathbf{X}_i]\right| > \frac{\epsilon}{1+\gamma} \,\middle|\, \mathbf{X}_i = \mathbf{x}\right\} \\
&+ \mathbf{P}\left\{\left|\frac{\gamma}{n_-}\sum_{j \in I_-} k_\sigma(\mathbf{X}_j, \mathbf{X}_i) - \gamma\mathbf{E}[k_\sigma(\mathbf{W}, \mathbf{X}_i) \mid \mathbf{X}_i]\right| > \frac{\gamma\epsilon}{1+\gamma} \,\middle|\, \mathbf{X}_i = \mathbf{x}\right\} \quad (7)
\end{aligned}
$$

By Hoeffding's inequality [16], the first term in (7) is

$$
\begin{aligned}
\mathbf{P}&\left\{\left|\sum_{j \in I_+, j \neq i} k_\sigma(\mathbf{X}_j, \mathbf{X}_i) - (n_+ - 1)\mathbf{E}[k_\sigma(\mathbf{Z}, \mathbf{X}_i) \mid \mathbf{X}_i]\right| > \frac{(n_+ - 1)\epsilon}{1+\gamma} \,\middle|\, \mathbf{X}_i = \mathbf{x}\right\} \\
&= \mathbf{P}\left\{\left|\sum_{j \in I_+, j \neq i} k_\sigma(\mathbf{X}_j, \mathbf{X}_i) - \mathbf{E}\left[\sum_{j \in I_+, j \neq i} k_\sigma(\mathbf{X}_j, \mathbf{X}_i) \mid \mathbf{X}_i\right]\right| > \frac{(n_+ - 1)\epsilon}{1+\gamma} \,\middle|\, \mathbf{X}_i = \mathbf{x}\right\} \\
&\leq 2e^{-2(n_+ - 1)\epsilon^2/(1+\gamma)^2 M^2}.
\end{aligned}
$$

The second term in (7) is

$$
\begin{aligned}
\mathbf{P}&\left\{\left|\sum_{j \in I_-} k_\sigma(\mathbf{X}_j, \mathbf{X}_i) - n_-\mathbf{E}[k_\sigma(\mathbf{W}, \mathbf{X}_i) \mid \mathbf{X}_i]\right| > \frac{n_-\epsilon}{1+\gamma} \,\middle|\, \mathbf{X}_i = \mathbf{x}\right\} \\
&= \mathbf{P}\left\{\left|\sum_{j \in I_-} k_\sigma(\mathbf{X}_j, \mathbf{X}_i) - \mathbf{E}\left[\sum_{j \in I_-} k_\sigma(\mathbf{X}_j, \mathbf{X}_i) \mid \mathbf{X}_i\right]\right| > \frac{n_-\epsilon}{1+\gamma} \,\middle|\, \mathbf{X}_i = \mathbf{x}\right\} \\
&\leq 2e^{-2n_-\epsilon^2/(1+\gamma)^2 M^2} \leq 2e^{-2(n_- - 1)\epsilon^2/(1+\gamma)^2 M^2}.
\end{aligned}
$$

Therefore,

$$
\begin{aligned}
\mathbf{P}\left\{\left|\widehat{h}_i - h(\mathbf{X}_i)\right| \geq \epsilon\right\} &= \mathbf{E}\left[\mathbf{P}\left\{\left|\widehat{h}_i - h(\mathbf{X}_i)\right| \geq \epsilon \,\middle|\, \mathbf{X}_i = \mathbf{X}\right\}\right] \\
&\leq 2e^{-2(n_+ - 1)\epsilon^2/(1+\gamma)^2 M^2} + 2e^{-2(n_- - 1)\epsilon^2/(1+\gamma)^2 M^2}.
\end{aligned}
$$

In a similar way, it can be shown that for $i \in I_-$,

$$\mathbf{P}\left\{\left|\widehat{h}_i - h(\mathbf{X}_i)\right| > \epsilon\right\} \leq 2e^{-2(n_+ - 1)\epsilon^2/(1+\gamma)^2 M^2} + 2e^{-2(n_- - 1)\epsilon^2/(1+\gamma)^2 M^2}.$$

Then,

$$\mathbf{P}\left\{\sup_{\boldsymbol{\alpha}\in A}\left|H_n\left(\boldsymbol{\alpha}\right)-H\left(\boldsymbol{\alpha}\right)\right|>\epsilon\right\}=\mathbf{P}\left\{\sup_{\boldsymbol{\alpha}\in A}\left|\sum_{i=1}^{n}\alpha_i Y_i\left(\widehat{h}_i-h\left(\mathbf{X}_i\right)\right)\right|>\epsilon\right\}$$

$$\leq\quad\mathbf{P}\left\{\sup_{\boldsymbol{\alpha}\in A}\sum_{i=1}^{n}\alpha_i\left|Y_i\right|\left|\widehat{h}_i-h\left(\mathbf{X}_i\right)\right|>\epsilon\right\}$$

$$=\quad\mathbf{P}\left\{\sup_{\boldsymbol{\alpha}\in A}\sum_{i\in I_+}\alpha_i\left|\widehat{h}_i-h\left(\mathbf{X}_i\right)\right|+\sum_{i\in I_-}\alpha_i\gamma\left|\widehat{h}_i-h\left(\mathbf{X}_i\right)\right|>\epsilon\right\}$$

$$\leq\quad\mathbf{P}\left\{\sup_{\boldsymbol{\alpha}\in A}\sum_{i\in I_+}\alpha_i\left|\widehat{h}_i-h\left(\mathbf{X}_i\right)\right|>\frac{\epsilon}{1+\gamma}\bigg|B\right\}+\mathbf{P}\left\{\sup_{\boldsymbol{\alpha}\in A}\sum_{i\in I_-}\alpha_i\gamma\left|\widehat{h}_i-h\left(\mathbf{X}_i\right)\right|>\frac{\gamma\epsilon}{1+\gamma}\bigg|B\right\}$$

$$=\quad\mathbf{P}\left\{\max_{i\in I_+}\left|\widehat{h}_i-h\left(\mathbf{X}_i\right)\right|>\frac{\epsilon}{1+\gamma}\bigg|B\right\}+\mathbf{P}\left\{\max_{i\in I_-}\left|\widehat{h}_i-h\left(\mathbf{X}_i\right)\right|>\frac{\epsilon}{1+\gamma}\bigg|B\right\}$$

$$=\quad\mathbf{P}\left\{\bigcup_{i\in I_+}\left\{\left|\widehat{h}_i-h\left(\mathbf{X}_i\right)\right|>\frac{\epsilon}{1+\gamma}\right\}\bigg|B\right\}+\mathbf{P}\left\{\bigcup_{i\in I_-}\left\{\left|\widehat{h}_i-h\left(\mathbf{X}_i\right)\right|>\frac{\epsilon}{1+\gamma}\right\}\bigg|B\right\}$$

$$\leq\quad\sum_{i\in I_+}\mathbf{P}\left\{\left|\widehat{h}_i-h\left(\mathbf{X}_i\right)\right|>\frac{\epsilon}{1+\gamma}\bigg|B\right\}+\sum_{i\in I_-}\mathbf{P}\left\{\left|\widehat{h}_i-h\left(\mathbf{X}_i\right)\right|>\frac{\epsilon}{1+\gamma}\bigg|B\right\}$$

$$\leq\quad n_+\left(2e^{-2(n_+-1)\epsilon^2/(1+\gamma)^4 M^2}+2e^{-2(n_--1)\epsilon^2/(1+\gamma)^4 M^2}\right)$$

$$+\,n_-\left(2e^{-2(n_+-1)\epsilon^2/(1+\gamma)^4 M^2}+2e^{-2(n_--1)\epsilon^2/(1+\gamma)^4 M^2}\right)$$

$$=\quad n\left(2e^{-2(n_+-1)\epsilon^2/(1+\gamma)^4 M^2}+2e^{-2(n_--1)\epsilon^2/(1+\gamma)^4 M^2}\right).$$

## A.2 Proof of Theorem 3

From Theorem 3 in [17], it suffices to show that

$$\int\left(\widehat{d}_\gamma\left(\mathbf{x};\widehat{\boldsymbol{\alpha}}\right)-d_{\gamma^*}\left(\mathbf{x}\right)\right)^2 d\mathbf{x}\to 0$$

in probability. Since from the triangle inequality

$$\|\widehat{d}_\gamma\left(\mathbf{x};\widehat{\boldsymbol{\alpha}}\right)-d_{\gamma^*}\left(\mathbf{x}\right)\|_{L^2}=\|\widehat{d}_\gamma\left(\mathbf{x};\widehat{\boldsymbol{\alpha}}\right)-d_\gamma\left(\mathbf{x}\right)+\left(\gamma-\gamma^*\right)f_-\left(\mathbf{x}\right)\|_{L^2}$$

$$\leq\|\widehat{d}_\gamma\left(\mathbf{x};\widehat{\boldsymbol{\alpha}}\right)-d_\gamma\left(\mathbf{x}\right)\|_{L_2}+\|\left(\gamma-\gamma^*\right)f_-\left(\mathbf{x}\right)\|_{L^2}$$

$$=\sqrt{ISE\left(\widehat{\boldsymbol{\alpha}}\right)}+\left|\gamma-\gamma^*\right|\cdot\|f_-\left(\mathbf{x}\right)\|_{L^2},$$

we need to show that $ISE\left(\widehat{\boldsymbol{\alpha}}\right)$ and $\gamma$ converges in probability to $0$ and $\gamma^*$, respectively. The convergence of $\gamma$ to $\gamma^*$ can be easily shown from the strong law of large numbers.

In the previous analyses, we have shown the convergence of $ISE\left(\widehat{\boldsymbol{\alpha}}\right)$ by treating $N_+, N_-$ and $\gamma$ as deterministic variables but now we turn to the case where these variables are random. Define an event $D=\left\{N_+\geq\frac{np}{2},N_-\geq\frac{n(1-p)}{2},\gamma\leq 2\gamma^*\right\}$. For any $\epsilon>0$,

$$\mathbf{P}\left\{ISE\left(\widehat{\boldsymbol{\alpha}}\right)>\epsilon\right\}\leq\mathbf{P}\left\{D^c\right\}+\mathbf{P}\left\{ISE\left(\widehat{\boldsymbol{\alpha}}\right)>\epsilon,D\right\}.$$

The first term converges to $0$ from the strong law of large numbers. Let define a set $S=\left\{(n_+,n_-)\,\big|\,n_+\geq\frac{np}{2},n_-\geq\frac{n(1-p)}{2},\frac{n_-}{n_+}\leq 2\gamma^*\right\}$. Then,

$$\mathbf{P}\left\{ISE\left(\widehat{\boldsymbol{\alpha}}\right)>\epsilon,D\right\}$$

$$=\quad\sum\mathbf{P}\left\{ISE\left(\widehat{\boldsymbol{\alpha}}\right)>\epsilon,D\,\big|\,N_+=n_+,N_-=n_-\right\}\cdot\mathbf{P}\left\{N_+=n_+,N_-=n_-\right\}$$

$$=\quad\sum_{(n_+,n_-)\in S}\mathbf{P}\left\{ISE\left(\widehat{\boldsymbol{\alpha}}\right)>\epsilon\,\big|\,N_+=n_+,N_-=n_-\right\}\cdot\mathbf{P}\left\{N_+=n_+,N_-=n_-\right\}$$

$$\leq\quad\max_{(n_+,n_-)\in S}\mathbf{P}\left\{ISE\left(\widehat{\boldsymbol{\alpha}}\right)>\epsilon\,\big|\,N_+=n_+,N_-=n_-\right\}.\tag{8}$$

Provided that $\sigma \to 0$ and $n\sigma^{2d}/\ln n \to \infty$, any pair $(n_+, n_-) \in S$ satisfies $\sigma \to 0$, $n_+\sigma^{2d}/\ln n \to \infty$, and $n_-\sigma^{2d}/\ln n \to \infty$ as $n \to \infty$ and thus the term in (8) converges to 0 from Theorem 2. This proves the theorem.

# References

[1] B. Schölkopf and A. J. Smola, *Learning with Kernels*, MIT Press, Cambridge, MA, 2002.

[2] C. Cortes and V. Vapnik, "Support-vector networks," *Machine Learning*, vol. 20, no. 3, pp. 273–297, 1995.

[3] J. Kim and C. Scott, "Kernel classification via integrated squared error," *IEEE Workshop on Statistical Signal Processing*, August 2007.

[4] D. Kim, *Least Squares Mixture Decomposition Estimation*, unpublished doctoral dissertation, Dept. of Statistics, Virginia Polytechnic Inst. and State Univ., 1995.

[5] Mark Girolami and Chao He, "Probability density estimation from optimally condensed data samples," *IEEE Transactions on Pattern Analysis and Machine Intelligence*, vol. 25, no. 10, pp. 1253–1264, OCT 2003.

[6] B.A. Turlach, "Bandwidth selection in kernel density estimation: A review," *Technical Report 9317, C.O.R.E. and Institut de Statistique, Université Catholique de Louvain*, 1993.

[7] David W.Scott, "Parametric statistical modeling by minimum integrated square error," *Technometrics 43*, pp. 274–285, 2001.

[8] A.B. Tsybakov F. Bunea and M.H. Wegkamp, "Sparse density estimation with $l_1$ penalties," *Proceedings of 20th Annual Conference on Learning Theory, COLT 2007, Lecture Notes in Artificial Intelligence, v4539*, pp. 530– 543, 2007.

[9] Ph. Rigollet and A.B. Tsybakov, "Linear and convex aggregation of density estimators," `https://hal.ccsd.cnrs.fr/ccsd-00068216`, 2004.

[10] Robert Jenssen, Deniz Erdogmus, Jose C.Principe, and Torbjørn Eltoft, "Towards a unification of information theoretic learning and kernel method," in *Proc. IEEE Workshop on Machine Learning for Signal Processing (MLSP2004),* Sao Luis, Brazil.

[11] Peter Hall and Matthew P.Wand, "On nonparametric discrimination using density differeces," *Biometrika*, vol. 75, no. 3, pp. 541–547, Sept 1988.

[12] P. Meinicke, T. Twellmann, and H. Ritter, "Discriminative densities from maximum contrast estimation," in *Advances in Neural Information Proceeding Systems 15,* Vancouver, Canada, 2002, pp. 985–992.

[13] M. Di Marzio and C.C. Taylor, "Kernel density classification and boosting: an $l_2$ analysis," *Statistics and Computing*, vol. 15, pp. 113–123(11), April 2005.

[14] E. Lehmann, *Testing statistical hypotheses*, Wiley, New York, 1986.

[15] M.P. Wand and M.C. Jones, *Kernel Smoothing*, Chapman & Hall, 1995.

[16] L. Devroye and G. Lugosi, "Combinatorial methods in density estimation," 2001.

[17] Charles T. Wolverton and Terry J. Wagner, "Asymptotically optimal discriminant fucntions for pattern classification," *IEEE Trans. Info. Theory*, vol. 15, no. 2, pp. 258–265, Mar 1969.

